# Spectral Learning of General Weighted Automata via Constrained Matrix Completion

**Borja Balle**
Universitat Politècnica de Catalunya
bballe@lsi.upc.edu

**Mehryar Mohri**
Courant Institute and Google Research
mohri@cims.nyu.edu

## Abstract

Many tasks in text and speech processing and computational biology require estimating functions mapping strings to real numbers. A broad class of such functions can be defined by weighted automata. Spectral methods based on the singular value decomposition of a Hankel matrix have been recently proposed for learning a probability distribution represented by a weighted automaton from a training sample drawn according to this same target distribution. In this paper, we show how spectral methods can be extended to the problem of learning a general weighted automaton from a sample generated by an arbitrary distribution. The main obstruction to this approach is that, in general, some entries of the Hankel matrix may be missing. We present a solution to this problem based on solving a constrained matrix completion problem. Combining these two ingredients, matrix completion and spectral method, a whole new family of algorithms for learning general weighted automata is obtained. We present generalization bounds for a particular algorithm in this family. The proofs rely on a joint stability analysis of matrix completion and spectral learning.

## 1 Introduction

Many tasks in text and speech processing, computational biology, or learning models of the environment in reinforcement learning, require estimating a function mapping variable-length sequences to real numbers. A broad class of such functions can be defined by weighted automata. The mathematical and algorithmic properties of weighted automata have been extensively studied in the most general setting where they are defined in terms of an arbitrary semiring [28, 9, 23]. Weighted automata are widely used in applications ranging from natural text and speech processing [24] to optical character recognition [12] and image processing [1]. This paper addresses the problem of learning weighted automata from a finite set of labeled examples.

The particular instance of this problem where the objective is to learn a probabilistic automaton from examples drawn from this same distribution has recently drawn much attention: starting with the seminal work of Hsu et al. [19], the so-called spectral method has proven to be a valuable tool in developing novel and theoretically-sound algorithms for learning HMMs and other related classes of distributions [5, 30, 31, 10, 6, 4]. Spectral methods have also been applied to other probabilistic models of practical interest, including probabilistic context-free grammars and graphical models with hidden variables [26, 22, 16, 3, 2]. The main idea behind these algorithms is that, under an identifiability assumption, the method of moments can be used to formulate a set of equations relating the parameters defining the target to observable statistics. Given enough training data, these statistics can be accurately estimated. Then, solving the corresponding approximate equations yields a model that closely estimates the target distribution. The *spectral* term takes its origin from the use of a singular value decomposition in solving those equations.

This paper tackles a significantly more general and more challenging problem than the specific instance just mentioned. Indeed, in general, there seems to be a large gap separating the scenario of learning a probabilistic automaton using data drawn according to the distribution it generates, from that of learning an arbitrary weighted automaton from labeled data drawn from some unknown distribution. For a start, in the former setting there is only one object to care about because the distribution from which examples are drawn *is* the target machine. In contrast, the latter involves two distinct objects: a distribution according to which strings are drawn, and a target weighted automaton assigning labels to these strings. It is not difficult in this setting to conceive that, for a particular target, an adversary could find a distribution over strings making the learner's task insurmountably difficult. In fact, this is the core idea behind the cryptography-based hardness results for learning deterministic finite automata given by Kearns and Valiant [20] – these same results apply to our setting as well.

But, even in cases where the distribution "cooperates," there is still an obstruction in leveraging the spectral method for learning general weighted automata. The statistics used by the spectral method are essentially the probabilities assigned by the target distribution to each string in some fixed finite set $\mathcal{B}$. In the case where the target is a distribution, increasingly large samples yield uniformly convergent estimates for these probabilities. Thus, it can be safely assumed that the probability of any string from $\mathcal{B}$ not present in the sample is zero. When learning arbitrary weighted automata, however, the value assigned by the target to an unseen string is unknown. Furthermore, one cannot expect that a sample would contain the values of the target function for all the strings in $\mathcal{B}$. This observation raises the question of whether it is possible at all to apply the spectral method in a setting with missing data, or, alternatively, whether there is a principled way to "estimate" this missing information and then apply the spectral method.

As it turns out, the latter approach can be naturally formulated as a constrained matrix completion problem. When applying the spectral method, the (approximate) values of the target on $\mathcal{B}$ are arranged in a matrix $\mathbf{H}$. Thus, the main difference between the two settings can be restated as follows: when learning a weighted automaton representing a distribution, unknown entries of $\mathbf{H}$ can be filled in with zeros, while in the general setting there is a priori no straightforward method to fill in the missing values. We propose to use a matrix completion algorithm for solving this last problem. In particular, since $\mathbf{H}$ is a Hankel matrix whose entries must satisfy some equality constraints, it turns out that the problem of learning weighted automata under an arbitrary distribution leads to what we call the *Hankel matrix completion* problem. This is essentially a *constrained* matrix completion problem where entries of valid hypotheses need to satisfy a set of equalities. We give an algorithm for solving this problem via convex optimization. Many existing approaches to matrix completion, e.g., [14, 13, 27, 18], are also based on convex optimization. Since the set of valid hypotheses for our constrained matrix completion problem is convex, many of these algorithms could also be modified to deal with the Hankel matrix completion problem.

In summary, our approach leverages two recent techniques for learning a general weighted automaton: matrix completion and spectral learning. It consists of first predicting the missing entries in $\mathbf{H}$ and then applying the spectral method to the resulting matrix. Altogether, this yields a family of algorithms parametrized by the choice of the specific Hankel matrix completion algorithm used. These algorithms are designed for learning an arbitrary weighted automaton from samples generated by an unknown distribution over strings and labels.

We study a special instance of this family of algorithms and prove generalization guarantees for its performance based on a stability analysis, under mild conditions on the distribution. The proof contains two main novel ingredients: a stability analysis of an algorithm for constrained matrix completion, and an extension of the analysis of spectral learning to an agnostic setting where data is generated by an arbitrary distribution and labeled by a process not necessarily modeled by a weighted automaton.

The rest of the paper is organized as follows. Section 2 introduces the main notation and definitions used in subsequent sections. In Section 3, we describe a family of algorithms for learning general weighted automata by combining constrained matrix completion and spectral methods. In Section 4, we give a detailed analysis of one particular algorithm in this family, including generalization bounds.

## 2 Preliminaries

This section introduces the main notation used in this paper. Bold letters will be used for vectors $\mathbf{v}$ and matrices $\mathbf{M}$. For vectors, $\|\mathbf{v}\|$ denotes the standard euclidean norm. For matrices, $\|\mathbf{M}\|$ denotes the operator norm. For $p \in [1, +\infty]$, $\|\mathbf{M}\|_p$ denotes the Schatten $p$-norm: $\|\mathbf{M}\|_p = (\sum_{n \geq 1} \sigma_n^p(\mathbf{M}))^{1/p}$, where $\sigma_n(\mathbf{M})$ is the $n$th singular value of $\mathbf{M}$. The special case $p = 2$ coincides with the Frobenius norm which will be sometimes also written as $\|\mathbf{M}\|_F$. The Moore–Penrose pseudo-inverse of a matrix $\mathbf{M}$ is denoted by $\mathbf{M}^+$.

### 2.1 Functions over Strings and Hankel Matrices

We denote by $\Sigma = \{a_1, \ldots, a_k\}$ a finite alphabet of size $k \geq 1$ and by $\epsilon$ the empty string. We also write $\Sigma' = \{\epsilon\} \cup \Sigma$. The set of all strings over $\Sigma$ is denoted by $\Sigma^\star$ and the length of a string $x$ denoted by $|x|$. For any $n \geq 0$, $\Sigma^{\leq n}$ denotes the set of all strings of length at most $n$. Given two sets of strings $\mathcal{P}, \mathcal{S} \subseteq \Sigma^\star$ we denote by $\mathcal{PS}$ the set of all strings $uv$ obtained by concatenation of a string $u \in \mathcal{P}$ and a string $v \in \mathcal{S}$. A set of strings $\mathcal{P}$ is called $\Sigma$-*complete* when $\mathcal{P} = \mathcal{P}'\Sigma'$ for some set $\mathcal{P}'$. $\mathcal{P}'$ is then called the *root* of $\mathcal{P}$. A pair $(\mathcal{P}, \mathcal{S})$ with $\mathcal{P}, \mathcal{S} \subseteq \Sigma^\star$ is said to form a *basis* of $\Sigma^\star$ if $\epsilon \in \mathcal{P} \cap \mathcal{S}$ and $\mathcal{P}$ is $\Sigma$-complete. We define the *dimension* of a basis $(\mathcal{P}, \mathcal{S})$ as the cardinality of $\mathcal{PS}$, that is $|\mathcal{PS}|$.

For any basis $\mathcal{B} = (\mathcal{P}, \mathcal{S})$, we denote by $\mathbb{H}_\mathcal{B}$ the vector space of functions $\mathbb{R}^{\mathcal{PS}}$ whose dimension is the dimension of $\mathcal{B}$. We will simply write $\mathbb{H}$ instead of $\mathbb{H}_\mathcal{B}$ when the basis $\mathcal{B}$ is clear from the context. The *Hankel matrix* $\mathbf{H} \in \mathbb{R}^{\mathcal{P} \times \mathcal{S}}$ associated to a function $h \in \mathbb{H}$ is the matrix whose entries are defined by $\mathbf{H}(u, v) = h(uv)$ for all $u \in \mathcal{P}$ and $v \in \mathcal{S}$. Note that the mapping $h \mapsto \mathbf{H}$ is linear. In fact, $\mathbb{H}$ is isomorphic to the vector space formed by all $|\mathcal{P}| \times |\mathcal{S}|$ real Hankel matrices and we can thus write by identification

$$\mathbb{H} = \left\{ \mathbf{H} \in \mathbb{R}^{\mathcal{P} \times \mathcal{S}} : \forall u_1, u_2 \in \mathcal{P}, \; \forall v_1, v_2 \in \mathcal{S}, \quad u_1 v_1 = u_2 v_2 \; \Rightarrow \; \mathbf{H}(u_1, v_1) = \mathbf{H}(u_2, v_2) \right\} \;.$$

It is clear from this characterization that $\mathbb{H}$ is a *convex* set because it is a subset of a convex space defined by equality constraints. In particular, a matrix in $\mathbb{H}$ contains $|\mathcal{P}||\mathcal{S}|$ coefficients with $|\mathcal{PS}|$ degrees of freedom, and the dependencies can be specified as a set of equalities of the form $\mathbf{H}(u_1, v_1) = \mathbf{H}(u_2, v_2)$ when $u_1 v_1 = u_2 v_2$. We will use both characterizations of $\mathbb{H}$ indistinctly for the rest of the paper. Also, note that different orderings of $\mathcal{P}$ and $\mathcal{S}$ may result in different sets of matrices. For convenience, we will assume for all that follows an arbitrary fixed ordering, since the choice of that order has no effect on any of our results.

Matrix norms extend naturally to norms in $\mathbb{H}$. For any $p \in [1, +\infty]$, the *Hankel–Schatten $p$-norm* on $\mathbb{H}$ is defined as $\|h\|_p = \|\mathbf{H}\|_p$. It is straightforward to verify that $\|h\|_p$ is a norm by the linearity of $h \mapsto \mathbf{H}$. In particular, this implies that the function $\|\cdot\|_p \colon \mathbb{H} \to \mathbb{R}$ is convex. In the case $p = 2$, it can be seen that $\|h\|_2^2 = \langle h, h \rangle_\mathbb{H}$, with the inner product on $\mathbb{H}$ defined by

$$\langle h, h' \rangle_\mathbb{H} = \sum_{x \in \mathcal{PS}} c_x h(x) h'(x) \;,$$

where $c_x = |\{(u, v) \in \mathcal{P} \times \mathcal{S} \colon x = uv\}|$ is the number of possible decompositions of $x$ into a prefix in $\mathcal{P}$ and a suffix in $\mathcal{S}$.

### 2.2 Weighted finite automata

A widely used class of functions mapping strings to real numbers is that of functions defined by *weighted finite automata* (WFA) or in short *weighted automata* [23]. These functions are also known as *rational power series* [28, 9]. A WFA over $\Sigma$ with $n$ states can be defined as a tuple $A = \langle \boldsymbol{\alpha}, \boldsymbol{\beta}, \{\mathbf{A}_a\}_{a \in \Sigma} \rangle$, where $\boldsymbol{\alpha}, \boldsymbol{\beta} \in \mathbb{R}^n$ are the *initial* and *final weight* vectors, and $\mathbf{A}_a \in \mathbb{R}^{n \times n}$ the transition matrix associated to each alphabet symbol $a \in \Sigma$. The function $f_A$ realized by a WFA $A$ is defined by

$$f_A(x) = \boldsymbol{\alpha}^\top \mathbf{A}_{x_1} \cdots \mathbf{A}_{x_t} \boldsymbol{\beta} \;,$$

for any string $x = x_1 \cdots x_t \in \Sigma^*$ with $t = |x|$ and $x_i \in \Sigma$ for all $i \in [1, t]$. We will say that a WFA $A = \langle \boldsymbol{\alpha}, \boldsymbol{\beta}, \{\mathbf{A}_a\} \rangle$ is $\gamma$-*bounded* if $\|\boldsymbol{\alpha}\|, \|\boldsymbol{\beta}\|, \|\mathbf{A}_a\| \leq \gamma$ for all $a \in \Sigma$. This property is convenient to bound the maximum value assigned by a WFA to any string of a given length.

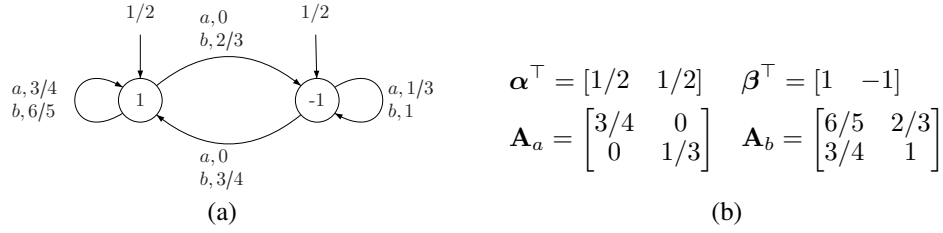

(a)                                                    (b)

Figure 1: Example of a weighted automaton over $\Sigma = \{a, b\}$ with 2 states: (a) graph representation; (b) algebraic representation.

WFAs can be more generally defined over an arbitrary semiring instead of the field of real numbers and are also known as *multiplicity automata* (e.g., [8]). To any function $f \colon \Sigma^\star \to \mathbb{R}$, we can associate its *Hankel matrix* $\mathbf{H}_f \in \mathbb{R}^{\Sigma^\star \times \Sigma^\star}$ with entries defined by $\mathbf{H}_f(u, v) = f(uv)$. These are just the bi-infinite versions of the Hankel matrices we introduced in the case $\mathcal{P} = \mathcal{S} = \Sigma^\star$. Carlyle and Paz [15] and Fliess [17] gave the following characterization of the set of functions $f$ in $\mathbb{R}^{\Sigma^\star}$ defined by a WFA in terms of the rank of their Hankel matrix $\mathrm{rank}(\mathbf{H}_f)$.[1]

**Theorem 1 ([15, 17])** *A function* $f \colon \Sigma^\star \to \mathbb{R}$ *can be defined by a WFA iff* $\mathrm{rank}(\mathbf{H}_f)$ *is finite and in that case* $\mathrm{rank}(\mathbf{H}_f)$ *is the minimal number of states of any WFA $A$ such that $f = f_A$.*

Thus, WFAs can be viewed as those functions whose Hankel matrix can be finitely "compressed". Since finite sub-blocks of a Hankel matrix cannot have a larger rank than its bi-infinite extension, this justifies the use of a low-rank-enforcing regularization in the definition of a Hankel matrix completion.

Note that *deterministic finite automata* (DFA) with $n$ states can be represented by a WFA with at most $n$ states. Thus, the results we present here can be directly applied to classification problems in $\Sigma^\star$. However, specializing our results to this particular setting may yield several improvements.

### 2.2.1 Example

Figure 1 shows an example of a weighted automaton $A = \langle \boldsymbol{\alpha}, \boldsymbol{\beta}, \{\mathbf{A}_a\} \rangle$ with two states defined over the alphabet $\Sigma = \{a, b\}$, with both its algebraic representation (Figure 1(b)) in terms of vectors and matrices and the equivalent graph representation (Figure 1(a)) useful for a variety of WFA algorithms [23]. Let $\mathcal{W} = \{\epsilon, a, b\}$, then $\mathcal{B} = (\mathcal{W}\Sigma', \mathcal{W})$ is a $\Sigma$-complete basis. The following is the Hankel matrix of $A$ on this basis shown with three-digit precision entries:

$$\mathbf{H}_\mathcal{B}^\top = \begin{bmatrix} & \epsilon & a & b & aa & ab & ba & bb \\ \epsilon & 0.00 & 0.20 & 0.14 & 0.22 & 0.15 & 0.45 & 0.31 \\ a & 0.20 & 0.22 & 0.45 & 0.19 & 0.29 & 0.45 & 0.85 \\ b & 0.14 & 0.15 & 0.31 & 0.13 & 0.20 & 0.32 & 0.58 \end{bmatrix}.$$

By Theorem 1, the Hankel matrix of $A$ has rank at most 2. Given $\mathbf{H}_\mathcal{B}$, the spectral method described in [19] can be used to recover a WFA $\hat{A}$ equivalent to $A$, in the sense that $A$ and $\hat{A}$ compute the same function. In general, one may be given a sample of strings labeled using some WFA that does not contain enough information to fully specify a Hankel matrix over a complete basis. In that case, Theorem 1 motivates the use of a low-rank matrix completion algorithm to fill in the missing entries in $\mathbf{H}_\mathcal{B}$ prior to the application of the spectral method. This is the basis of the algorithm we describe in the following section.

## 3 The HMC+SM Algorithm

In this section we describe our algorithm HMC+SM for learning weighted automata. As input, the algorithm takes a sample $Z = (z_1, \ldots, z_m)$ containing $m$ examples $z_i = (x_i, y_i) \in \Sigma^\star \times \mathbb{R}$,

$1 \leq i \leq m$, drawn i.i.d. from some distribution $\mathcal{D}$ over $\Sigma^\star \times \mathbb{R}$. There are three parameters a user can specify to control the behavior of the algorithm: a basis $\mathcal{B} = (\mathcal{P}, \mathcal{S})$ of $\Sigma^\star$, a regularization parameter $\tau > 0$, and the desired number of states $n$ in the hypothesis. The output returned by HMC+SM is a WFA $A_Z$ with $n$ states that computes a function $f_{A_Z} \colon \Sigma^\star \to \mathbb{R}$.

The algorithm works in two stages. In the first stage, a constrained matrix completion algorithm with input $Z$ and regularization parameter $\tau$ is used to return a Hankel matrix $\mathbf{H}_Z \in \mathbb{H}_{\mathcal{B}}$. In the second stage, the spectral method is applied to $\mathbf{H}_Z$ to compute a WFA $A_Z$ with $n$ states. These two steps will be described in detail in the following sections.

As will soon become apparent, HMC+SM defines in fact a whole family of algorithms. In particular, by combining the spectral method with any algorithm for solving the *Hankel matrix completion problem*, one can derive a new algorithm for learning WFAs. For concreteness, in the following, we will only consider the Hankel matrix completion algorithm described in Section 3.1. Through its parametrization by a number $1 \leq p \leq \infty$ and a convex loss $\ell \colon \mathbb{R} \times \mathbb{R} \to \mathbb{R}_+$, this completion algorithm already gives rise to a family of learning algorithms that we denote by $\mathsf{HMC}_{\mathsf{p},\ell}+\mathsf{SM}$. However, it is important to keep in mind that for each existing matrix completion algorithm that can be modified to solve the Hankel matrix completion problem, a new algorithm for learning WFAs can be obtained via the general scheme we describe below.

## 3.1 Hankel Matrix Completion

We now describe our Hankel matrix completion algorithm. Given a basis $\mathcal{B} = (\mathcal{P}, \mathcal{S})$ of $\Sigma^\star$ and a sample $Z$ over $\Sigma^\star \times \mathbb{R}$, the algorithm solves a convex optimization problem and returns a matrix $\mathbf{H}_Z \in \mathbb{H}_{\mathcal{B}}$. We give two equivalent descriptions of this optimization, one in terms of functions $h \colon \mathcal{PS} \to \mathbb{R}$, and another in terms of Hankel matrices $\mathbf{H} \in \mathbb{R}^{\mathcal{P} \times \mathcal{S}}$. While the former is perhaps conceptually simpler, the latter is easier to implement within the existing frameworks of convex optimization.

We will denote by $\widetilde{Z}$ the subsample of $Z$ formed by examples $z = (x, y)$ with $x \in \mathcal{PS}$ and by $\widetilde{m}$ its size $|\widetilde{Z}|$. For any $p \in [1, +\infty]$ and a convex loss function $\ell \colon \mathbb{R} \times \mathbb{R} \to \mathbb{R}_+$, we consider the objective function $F_Z$ defined for any $h \in \mathbb{H}$ by

$$F_Z(h) = \tau N(h) + \widehat{R}_{\widetilde{Z}}(h) = \tau \|h\|_p^2 + \frac{1}{\widetilde{m}} \sum_{(x,y) \in \widetilde{Z}} \ell(h(x), y) \ ,$$

where $\tau > 0$ is a regularization parameter. $F_Z$ is a convex function, by the convexity of $\|\cdot\|_p$ and $\ell$. Our algorithm seeks to minimize this loss function over the finite-dimensional vector space $\mathbb{H}$ and returns a function $h_Z$ satisfying

$$h_Z \in \operatorname*{argmin}_{h \in \mathbb{H}} F_Z(h) \ . \tag{HMC-h}$$

To define an equivalent optimization over the matrix version of $\mathbb{H}$, we introduce the following notation. For each string $x \in \mathcal{PS}$, fix a pair of coordinate vectors $(\mathbf{u}_x, \mathbf{v}_x) \in \mathbb{R}^{\mathcal{P}} \times \mathbb{R}^{\mathcal{S}}$ such that $\mathbf{u}_x^\top \mathbf{H} \mathbf{v}_x = \mathbf{H}(x)$ for any $\mathbf{H} \in \mathbb{H}$. That is, $\mathbf{u}_x$ and $\mathbf{v}_x$ are coordinate vectors corresponding respectively to a prefix $u \in \mathcal{P}$ and a suffix $v \in \mathcal{S}$, and such that $uv = x$. Now, abusing our previous notation, we define the following loss function over matrices:

$$F_Z(\mathbf{H}) = \tau N(\mathbf{H}) + \widehat{R}_{\widetilde{Z}}(\mathbf{H}) = \tau \|\mathbf{H}\|_p^2 + \frac{1}{\widetilde{m}} \sum_{(x,y) \in \widetilde{Z}} \ell(\mathbf{u}_x^\top \mathbf{H} \mathbf{v}_x, y) \ .$$

This is a convex function defined over the space of all $|\mathcal{P}| \times |\mathcal{S}|$ matrices. Optimizing $F_Z$ over the convex set of Hankel matrices $\mathbb{H}$ leads to an algorithm equivalent to (HMC-h):

$$\mathbf{H}_Z \in \operatorname*{argmin}_{\mathbf{H} \in \mathbb{H}} F_Z(H) \ . \tag{HMC-H}$$

We note here that our approach shares some common aspects with some previous work in matrix completion. The fact that there may not be a true underlying Hankel matrix makes it somewhat close to the agnostic setting in [18], where matrix completion is also applied under arbitrary distributions. Nonetheless, it is also possible to consider other learning frameworks for WFAs where algorithms for exact matrix completion [14, 27] or noisy matrix completion [13] may be useful. Furthermore, since most algorithms in the literature of matrix completion are based on convex optimization problems, it is likely that most of them can be adapted to solve constrained matrix completions problems such as the one we discuss here.

## 3.2 Spectral Method for General WFA

Here, we describe how the spectral method can be applied to $\mathbf{H}_Z$ to obtain a WFA. We use the same notation as in [7] and a version of the spectral method working with an arbitrary basis (as in [5, 4, 7]), in contrast to versions restricted to $\mathcal{P} = \Sigma^{\leq 2}$ and $\mathcal{S} = \Sigma$ like [19].

We first need to partition $\mathbf{H}_Z$ into $k + 1$ blocks as follows. Since $\mathcal{B}$ is a basis, $\mathcal{P}$ is $\Sigma$-complete and admits a root $\mathcal{P}'$. We define a block $\mathbf{H}_a \in \mathbb{R}^{\mathcal{P}' \times \mathcal{S}}$ for each $a \in \Sigma'$, whose entries are given by $\mathbf{H}_a(u, v) = \mathbf{H}_Z(ua, v)$, for any $u \in \mathcal{P}'$ and $v \in \mathcal{S}$. Thus, after suitably permuting the rows of $\mathbf{H}_Z$, we can write $\mathbf{H}_Z^\top = [\mathbf{H}_\epsilon^\top, \mathbf{H}_{a_1}^\top, \dots, \mathbf{H}_{a_k}^\top]$. We will use the following specific notation to refer to the rows and columns of $\mathbf{H}_\epsilon$ corresponding to $\epsilon \in \mathcal{P}' \cap \mathcal{S}$: $\mathbf{h}_{\epsilon,\mathcal{S}} \in \mathbb{R}^{\mathcal{S}}$ with $\mathbf{h}_{\epsilon,\mathcal{S}}(v) = \mathbf{H}_\epsilon(\epsilon, v)$ and $\mathbf{h}_{\mathcal{P}',\epsilon}(u) \in \mathbb{R}^{\mathcal{P}'}$ with $\mathbf{h}_{\mathcal{P}',\epsilon}(u) = \mathbf{H}_\epsilon(u, \epsilon)$.

Using this notation, the spectral method can be described as follows. Given the desired number of states $n$, it consists of first computing the truncated SVD of $\mathbf{H}_\epsilon$ corresponding to the $n$ largest singular values: $\mathbf{U}_n \mathbf{D}_n \mathbf{V}_n^\top$. Thus, matrix $\mathbf{U}_n \mathbf{D}_n \mathbf{V}_n^\top$ is the best rank $n$ approximation to $\mathbf{H}_\epsilon$ with respect to the Frobenius norm. Then, using the right singular vectors $\mathbf{V}_n$ of $\mathbf{H}_\epsilon$, the next step consists of computing a weighted automaton $A_Z = \langle \boldsymbol{\alpha}, \boldsymbol{\beta}, \{\mathbf{A}_a\} \rangle$ as follows:

$$\boldsymbol{\alpha}^\top = \mathbf{h}_{\epsilon,\mathcal{S}}^\top \mathbf{V}_n \qquad \boldsymbol{\beta} = (\mathbf{H}_\epsilon \mathbf{V}_n)^+ \mathbf{h}_{\mathcal{P}',\epsilon} \qquad \mathbf{A}_a = (\mathbf{H}_\epsilon \mathbf{V}_n)^+ \mathbf{H}_a \mathbf{V}_n \ . \qquad \text{(SM)}$$

The fact that the spectral method is based on a singular value decomposition justifies in part the use of a Schatten $p$-norm as a regularizer in (HMC-H). In particular, two very natural choices are $p = 1$ and $p = 2$. The first one corresponds to a nuclear norm regularized optimization, which is known to enforce a low rank constraint on $\mathbf{H}_Z$. In a sense, this choice can be justified in view of Theorem 1 when the target is known to be generated by some WFA. On the other hand, choosing $p = 2$ also has some effect on the spread of singular values, while at the same time enforcing the coefficients in $\mathbf{H}_Z$ – especially those that are completely unknown – to be small. As our analysis suggests, this last property is important for preventing errors from accumulating on the values assigned by $A_Z$ to long strings.

## 4 Generalization Bound

In this section, we study the generalization properties of $\mathsf{HMC}_{\mathsf{p},\ell} + \mathsf{SM}$. We give a stability analysis for a special instance of this family of algorithms and use it to derive a generalization bound. We study the specific case where $p = 2$ and $\ell(y, y') = |y - y'|$ for all $(y, y')$. But, much of our analysis can be used to derive similar bounds for other instances of $\mathsf{HMC}_{\mathsf{p},\ell} + \mathsf{SM}$. The proofs of the technical results presented are given in the Appendix.

We first introduce some notation needed for the presentation of our main result. For any $\nu > 0$, let $t_\nu$ be the function defined by $t_\nu(x) = x$ for $|x| \leq \nu$ and $t_\nu(x) = \nu \operatorname{sign}(x)$ for $|x| > \nu$. For any distribution $\mathcal{D}$ over $\Sigma^\star \times \mathbb{R}$, we denote by $\mathcal{D}_\Sigma$ its marginal distribution over $\Sigma^\star$. The probability that a string $x \sim \mathcal{D}_\Sigma$ belongs to $\mathcal{P}\mathcal{S}$ is denoted by $\pi = \mathcal{D}_\Sigma(\mathcal{P}\mathcal{S})$.

We assume that the parameters $\mathcal{B}$, $n$, and $\tau$ are fixed. Two parameters that depend on $\mathcal{D}$ will appear in our bound. In order to define these parameters, we need to consider the output $\mathbf{H}_Z$ of (HMC-H) as a random variable that depends on the sample $Z$. Writing $\mathbf{H}_Z^\top = [\mathbf{H}_\epsilon^\top, \mathbf{H}_{a_1}^\top, \dots, \mathbf{H}_{a_k}^\top]$, as in Section 3.2, we define:

$$\sigma = \mathop{\mathbb{E}}_{Z \sim \mathcal{D}^m} [\sigma_n(\mathbf{H}_\epsilon)] \qquad \rho = \mathop{\mathbb{E}}_{Z \sim \mathcal{D}^m} \left[ \sigma_n(\mathbf{H}_\epsilon)^2 - \sigma_{n+1}(\mathbf{H}_\epsilon)^2 \right] \ ,$$

where $\sigma_n(\mathbf{M})$ denotes the $n$th singular value of matrix $\mathbf{M}$. Note that these parameters may vary with $m$, $n$, $\tau$ and $\mathcal{B}$.

In contrast to previous learning results based on the spectral method, our bound holds in an agnostic setting. That is, we do not require that the data was generated from some (probabilistic) unknown WFA. However, in order to prove our results we do need to make two assumptions about the tails of the distribution. First, we need to assume that there exists a bound on the magnitude of the labels generated by the distribution.

**Assumption 1** *There exists a constant $\nu > 0$ such that if $(x, y) \sim \mathcal{D}$, then $|y| \leq \nu$ almost surely.*

Second, we assume that the strings generated by the distribution will not be too long. In particular, that the length of the strings generated by $\mathcal{D}_\Sigma$ follows a distribution whose tail is slightly lighter than sub-exponential.

**Assumption 2** *There exist constants $c, \eta > 0$ such that $\mathbb{P}_{x \sim \mathcal{D}_\Sigma}[|x| \geq t] \leq \exp(-ct^{1+\eta})$ holds for all $t \geq 0$.*

We note that in the present context both assumptions are quite reasonable. Assumption 1 is equivalent to assumptions made in other contexts where a stability analysis is pursued, e.g., in the analysis of support vector regression in [11]. Furthermore, in our context, this assumption can be relaxed to require only that the distribution over labels be sub-Gaussian, at the expense of a more complex proof.

Assumption 2 is required by the fact already pointed out in [19] that errors in the estimation of operator models accumulate exponentially with the length of the string. Moreover, it is well known that the tail of any probability distribution generated by a WFA is sub-exponential. Thus, though we do not require $\mathcal{D}_\Sigma$ to be generated by a WFA, we do need its distribution over lengths to have a tail behavior similar to that of a distribution generated by a WFA. This seems to be a limitation common to all known learnability proofs based on the spectral method.

We can now state our main result, which is a bound on the *average loss* $R(f) = \mathbb{E}_{z \sim \mathcal{D}}[\ell(f(x), y)]$ in terms of the *empirical loss* $\widehat{R}_Z(f) = |Z|^{-1} \sum_{z \in Z} \ell(f(x), y)$.

**Theorem 2** *Let $Z$ be a sample formed by $m$ i.i.d. examples generated from some distribution $\mathcal{D}$ satisfying Assumptions 1 and 2. Let $A_Z$ be the WFA returned by algorithm $\mathsf{HMC}_{\mathsf{p},\ell} + \mathsf{SM}$ with $p = 2$ and loss function $\ell(y, y') = |y - y'|$. Then, for any $\delta > 0$, the following holds with probability at least $1 - \delta$ for $f_Z = t_\nu \circ f_{A_Z}$:*

$$R(f_Z) \leq \widehat{R}_Z(f_Z) + O\left( \frac{\nu^4 |\mathcal{P}|^2 |\mathcal{S}|^{3/2}}{\tau \sigma^3 \rho \pi} \frac{\ln m}{m^{1/3}} \sqrt{\ln \frac{1}{\delta}} \right) \ .$$

The proof of this theorem is based on an algorithmic stability analysis. Thus, we will consider two samples of size $m$, $Z \sim \mathcal{D}^m$ consisting of $m$ i.i.d. examples drawn from $\mathcal{D}$, and $Z'$ differing from $Z$ by just one point: say $z_m$ in $Z = (z_1, \ldots, z_m)$ and $z'_m$ in $Z' = (z_1, \ldots, z_{m-1}, z'_m)$. The new example $z'_m$ is an arbitrary point the support of $\mathcal{D}$. Throughout the analysis we use the shorter notation $\mathbf{H} = \mathbf{H}_Z$ and $\mathbf{H}' = \mathbf{H}_{Z'}$ for the Hankel matrices obtained from (HMC-H) based on samples $Z$ and $Z'$ respectively.

The first step in the analysis is to bound the stability of the matrix completion algorithm. This is done in the following lemma, that gives a sample-dependent and a sample-independent bound for the stability of $\mathbf{H}$.

**Lemma 3** *Suppose $\mathcal{D}$ satisfies Assumption 1. Then, the following holds:*

$$\|\mathbf{H} - \mathbf{H}'\|_F \leq \min \left\{ 2\nu \sqrt{|\mathcal{P}||\mathcal{S}|}, \frac{1}{\tau \min\{\widetilde{m}, \widetilde{m}'\}} \right\} \ .$$

The standard method for deriving generalization bounds from algorithmic stability results could be applied here to obtain a generalization bound for our Hankel matrix completion algorithm. However, our goal is to give a generalization bound for the full $\mathsf{HMC} + \mathsf{SM}$ algorithm.

Using the bound on the Frobenius norm $\|\mathbf{H} - \mathbf{H}'\|_F$, we are able to analyze the stability of $\sigma_n(\mathbf{H}_\epsilon)$, $\sigma_n(\mathbf{H}_\epsilon)^2 - \sigma_{n+1}(\mathbf{H}_\epsilon)^2$, and $\mathbf{V}_n$ using well-known results on the stability of singular values and singular vectors. These results are used to bound the difference between the operators of WFA $A_Z$ and $A_{Z'}$. The following lemma can be proven by modifying and extending some of the arguments of [19, 4], which were given in the specific case of WFAs representing a probability distribution.

**Lemma 4** *Let $\varepsilon = \|\mathbf{H} - \mathbf{H}'\|_F$, $\widehat{\sigma} = \min\{\sigma_n(\mathbf{H}_\epsilon), \sigma_n(\mathbf{H}'_\epsilon)\}$, and $\widehat{\rho} = \sigma_n(\mathbf{H}_\epsilon)^2 - \sigma_{n+1}(\mathbf{H}_\epsilon)^2$. Suppose $\varepsilon \leq \sqrt{\widehat{\rho}}/4$. Then, there exists some constant $C > 0$ such that the following three inequalities*

*hold:*

$$\forall a \in \Sigma: \quad \|\mathbf{A}_a - \mathbf{A}'_a\| \leq C\varepsilon\nu^3 |\mathcal{P}|^{3/2} |\mathcal{S}|^{1/2} / \widehat{\rho}\widehat{\sigma}^2;$$
$$\|\boldsymbol{\alpha} - \boldsymbol{\alpha}'\| \leq C\varepsilon\nu^2 |\mathcal{P}|^{1/2} |\mathcal{S}| / \widehat{\rho};$$
$$\|\boldsymbol{\beta} - \boldsymbol{\beta}'\| \leq C\varepsilon\nu^3 |\mathcal{P}|^{3/2} |\mathcal{S}|^{1/2} / \widehat{\rho}\widehat{\sigma}^2.$$

The other half of the proof results from combining Lemmas 3 and 4 to obtain a bound for $|f_Z(x) - f_{Z'}(x)|$. This is a delicate step, because some of the bounds given above involve quantities that are defined in terms of $Z$. Therefore, all these parameters need to be controlled in order to ensure that the bounds do not grow too large. Furthermore, to obtain the desired bounds we need to extend the usual tools for analyzing spectral methods to the current setting. In particular, these tools need to be adapted to the agnostic settings where there is no underlying true WFA. The analysis is further complicated by the fact that now the functions we are trying to learn and the distribution that generates the data are not necessarily related.

Once all this is achieved, it remains to combine these new tools to show an algorithmic stability result for $\mathsf{HMC}_{\mathsf{p},\ell} + \mathsf{SM}$. In the following lemma, we first define "bad" samples $Z$ and show that bad samples have a very low probability.

**Lemma 5** *Suppose $\mathcal{D}$ satisfies Assumptions 1 and 2. If $Z$ is a large enough i.i.d. sample from $\mathcal{D}$, then with probability at least $1 - 1/m^3$ the following inequalities hold simultaneously: $|x_i| \leq ((1/c)\ln(4m^4))^{1/1+\eta}$ for all $i$, $\varepsilon \leq 4/(\tau\pi m)$, $\widehat{\sigma} \geq \sigma/2$, and $\widehat{\rho} \geq \rho/2$.*

After that we give two upper bounds for $|f_Z(x) - f_{Z'}(x)|$: a tighter bound that holds for "good" samples $Z$ and $Z'$ and a another one that holds for all samples. These bounds are combined using a variant of McDiarmid's inequality for dealing with functions that do not satisfy the bounded differences assumption almost surely [21]. The rest of the proof then follows the same scheme as the standard one for deriving generalization bounds for stable algorithms [11, 25].

## 5 Conclusion

We described a new algorithmic solution for learning arbitrary weighted automata from a sample of labeled strings drawn from an unknown distribution. Our approach combines an algorithm for constrained matrix completion with the recently developed spectral learning methods for learning probabilistic automata. Using our general scheme, a broad family of algorithms for learning weighted automata can be obtained. We gave a stability analysis of a particular algorithm in that family and used it to prove generalization bounds that hold for all distributions satisfying two reasonable assumptions. The particular case of Schatten $p$-norm with $p = 1$, which corresponds to a regularization with the nuclear norm, can be analyzed using similar techniques. Our results can be further extended by deriving generalization guarantees for all algorithms in the family we introduced. An extensive and rigorous empirical comparison of all these algorithms will be an important complement to the research we presented. Finally, learning DFAs under an arbitrary distribution using the algorithms we presented deserves a specific study since the problem is of interest in many applications and since it may benefit from improved learning guarantees.

## Acknowledgments

Borja Balle is partially supported by an FPU fellowship (AP2008-02064) and project TIN2011-27479-C04-03 (BASMATI) of the Spanish Ministry of Education and Science, the EU PASCAL2 NoE (FP7-ICT-216886), and by the Generalitat de Catalunya (2009-SGR-1428). The work of Mehryar Mohri was partly funded by the NSF grant IIS-1117591.

## Footnotes

[1]The construction of an equivalent WFA with the minimal number of states from a given WFA was first given by Schützenberger [29].

# References

[1] J. Albert and J. Kari. Digital image compression. In *Handbook of Weighted Automata*. Springer, 2009.

[2] A. Anandkumar, D. P. Foster, D. Hsu, S. M. Kakade, and Y-K. Liu. Two SVDs suffice: Spectral decompositions for probabilistic topic modeling and latent dirichlet allocation. *CoRR*, abs/1204.6703, 2012.

[3] A. Anandkumar, D. Hsu, and S. M. Kakade. A method of moments for mixture models and hidden Markov models. *COLT*, 2012.

[4] R. Bailly. Quadratic weighted automata: Spectral algorithm and likelihood maximization. *ACML*, 2011.

[5] R. Bailly, F. Denis, and L. Ralaivola. Grammatical inference as a principal component analysis problem. *ICML*, 2009.

[6] B. Balle, A. Quattoni, and X. Carreras. A spectral learning algorithm for finite state transducers. *ECML–PKDD*, 2011.

[7] B. Balle, A. Quattoni, and X. Carreras. Local loss optimization in operator models: A new insight into spectral learning. *ICML*, 2012.

[8] A. Beimel, F. Bergadano, N.H. Bshouty, E. Kushilevitz, and S. Varricchio. Learning functions represented as multiplicity automata. *JACM*, 2000.

[9] J. Berstel and C. Reutenauer. *Rational Series and Their Languages*. Springer, 1988.

[10] B. Boots, S. Siddiqi, and G. Gordon. Closing the learning planning loop with predictive state representations. *I. J. Robotic Research*, 2011.

[11] O. Bousquet and A. Elisseeff. Stability and generalization. *JMLR*, 2002.

[12] T. M. Breuel. The OCRopus open source OCR system. *IS&T/SPIE Annual Symposium*, 2008.

[13] E.J. Candes and Y. Plan. Matrix completion with noise. *Proceedings of the IEEE*, 2010.

[14] E.J. Candes and T. Tao. The power of convex relaxation: Near-optimal matrix completion. *IEEE Transactions on Information Theory*, 2010.

[15] Jack W. Carlyle and Azaria Paz. Realizations by stochastic finite automata. *J. Comput. Syst. Sci.*, 5(1):26–40, 1971.

[16] S. B. Cohen, K. Stratos, M. Collins, D. P. Foster, and L. Ungar. Spectral learning of latent-variable PCFGs. *ACL*, 2012.

[17] M. Fliess. Matrices de Hankel. *Journal de Mathématiques Pures et Appliquées*, 53:197–222, 1974.

[18] R. Foygel, R. Salakhutdinov, O. Shamir, and N. Srebro. Learning with the weighted trace-norm under arbitrary sampling distributions. *NIPS*, 2011.

[19] D. Hsu, S. M. Kakade, and T. Zhang. A spectral algorithm for learning hidden Markov models. *COLT*, 2009.

[20] M. Kearns and L. Valiant. Cryptographic limitations on learning boolean formulae and finite automata. *JACM*, 1994.

[21] S. Kutin. Extensions to McDiarmid's inequality when differences are bounded with high probability. Technical report, TR-2002-04, University of Chicago, 2002.

[22] F.M. Luque, A. Quattoni, B. Balle, and X. Carreras. Spectral learning in non-deterministic dependency parsing. *EACL*, 2012.

[23] M. Mohri. Weighted automata algorithms. In *Handbook of Weighted Automata*. Springer, 2009.

[24] M. Mohri, F. C. N. Pereira, and M. Riley. Speech recognition with weighted finite-state transducers. In *Handbook on Speech Processing and Speech Communication*. Springer, 2008.

[25] M. Mohri, A. Rostamizadeh, and A. Talwalkar. *Foundations of Machine Learning*. The MIT Press, 2012.

[26] A.P. Parikh, L. Song, and E.P. Xing. A spectral algorithm for latent tree graphical models. *ICML*, 2011.

[27] B. Recht. A simpler approach to matrix completion. *JMLR*, 2011.

[28] Arto Salomaa and Matti Soittola. *Automata-Theoretic Aspects of Formal Power Series*. Springer-Verlag: New York, 1978.

[29] M.P. Schützenberger. On the definition of a family of automata. *Information and Control*, 1961.

[30] S. M. Siddiqi, B. Boots, and G. J. Gordon. Reduced-rank hidden Markov models. *AISTATS*, 2010.

[31] L. Song, B. Boots, S. Siddiqi, G. Gordon, and A. Smola. Hilbert space embeddings of hidden Markov models. *ICML*, 2010.

